# Learning hierarchical structures with Linear Relational Embedding

**Alberto Paccanaro**    **Geoffrey E. Hinton**
Gatsby Computational Neuroscience Unit
UCL, 17 Queen Square, London, UK
{alberto,hinton}@gatsby.ucl.ac.uk

## Abstract

We present Linear Relational Embedding (LRE), a new method of learning a distributed representation of concepts from data consisting of instances of relations between given concepts. Its final goal is to be able to generalize, i.e. infer new instances of these relations among the concepts. On a task involving family relationships we show that LRE can generalize better than any previously published method. We then show how LRE can be used effectively to find compact distributed representations for variable-sized recursive data structures, such as trees and lists.

## 1   Linear Relational Embedding

Our aim is to take a large set of facts about a domain expressed as tuples of arbitrary symbols in a simple and rigid syntactic format and to be able to infer other "common-sense" facts without having any prior knowledge about the domain. Let us imagine a situation in which we have a set of concepts and a set of relations among these concepts, and that our data consists of few instances of these relations that hold among the concepts. We want to be able to infer other instances of these relations. For example, if the concepts are the people in a certain family, the relations are kinship relations, and we are given the facts "Alberto has-father Pietro" and "Pietro has-brother Giovanni", we would like to be able to infer "Alberto has-uncle Giovanni". Our approach is to learn appropriate distributed representations of the entities in the data, and then exploit the generalization properties of the distributed representations [2] to make the inferences. In this paper we present a method, which we have called Linear Relational Embedding (LRE), which learns a distributed representation for the concepts by embedding them in a space where the relations between concepts are linear transformations of their distributed representations.

Let us consider the case in which all the relations are binary, i.e. involve two concepts. In this case our data consists of triplets $(concept_1, relation, concept_2)$, and the problem we are trying to solve is to infer missing triplets when we are given only few of them. Inferring a triplet is equivalent to being able to complete it, that is to come up with one of its elements, given the other two. Here we shall always try to complete the third element of the triplets [1]. LRE will then represent each concept in the data as a learned vector in a

Euclidean space and each relationship between the two concepts as a learned matrix that maps the first concept into an approximation to the second concept. Let us assume that our data consists of $C$ such triplets containing $N$ distinct concepts and $M$ binary relations. We shall call this set of triplets $\mathcal{C}$; $\mathcal{V} = \{\mathbf{v}_1, \ldots, \mathbf{v}_N\}$ will denote the set of $n$-dimensional vectors corresponding to the $N$ concepts, and $\mathcal{R} = \{R_1, \ldots, R_M\}$ the set of $(n \times n)$ matrices corresponding to the $M$ relations. Often we shall need to indicate the vectors and the matrix which correspond to the concepts and the relation in a certain triplet $c$. In this case we shall denote the vector corresponding to the first concept with $\mathbf{a}$, the vector corresponding to the second concept with $\mathbf{b}$ and the matrix corresponding to the relation with $R$. We shall therefore write the triplet $c$ as $(\mathbf{a}^c, R^c, \mathbf{b}^c)$ where $\mathbf{a}^c, \mathbf{b}^c \in \mathcal{V}$ and $R^c \in \mathcal{R}$. The operation that relates a pair $(\mathbf{a}^c, R^c)$ to a vector $\mathbf{b}^c$ is the matrix-vector multiplication, $R^c \cdot \mathbf{a}^c$, which produces an approximation to $\mathbf{b}^c$. If for every triplet $(\mathbf{a}^c, R^c, \mathbf{b}^c)$ we think of $R^c \cdot \mathbf{a}^c$ as a noisy version of one of the concept vectors, then one way to learn an embedding is to maximize the probability that it is a noisy version of the correct completion, $\mathbf{b}^c$. We imagine that a concept has an average location in the space, but that each "observation" of the concept is a noisy realization of this average location. Assuming spherical Gaussian noise with a variance of $1/2$ on each dimension, the discriminative goodness function that corresponds to the log probability of getting the right completion, summed over all training triplets is:

$$D = \sum_{c=1}^{C} \frac{1}{k_c} \log \frac{e^{-||R^c \cdot \mathbf{a}^c - \mathbf{b}^c||^2}}{\sum_{\mathbf{v}_i \in \mathcal{V}} e^{-||R^c \cdot \mathbf{a}^c - \mathbf{v}_i||^2}} \tag{1}$$

where $k_c$ is the number of triplets in $\mathcal{C}$ having the first two terms equal to the ones of $c$, but differing in the third term [2].

Learning based on maximizing $D$ with respect to all the vector and matrix components has given good results, and has proved successful in generalization as well [5]. However, when we learn an embedding by maximizing $D$, we are not making use of exactly the information that we have in the triplets. For each triplet $c$, we are making the vector representing the correct completion $\mathbf{b}^c$ *more probable* than any other concept vector given $R^c \cdot \mathbf{a}^c$, while the triplet states that $R^c \cdot \mathbf{a}^c$ must be *equal* to $\mathbf{b}^c$. The numerator of $D$ does exactly this, but we also have the denominator, which is necessary in order to stay away from the trivial $\mathbf{0}$ solution [3]. We noticed however that the denominator is critical at the beginning of the learning, but as the vectors and matrices differentiate we could gradually lift this burden, allowing $\sum_{c=1}^{C} ||R^c \cdot \mathbf{a}^c - \mathbf{b}^c||^2$ to become the real goal of the learning. To do this we modify the discriminative function to include a parameter $\alpha$, which is annealed from 1 to 0 during learning [4]:

$$G = \sum_{c=1}^{C} \frac{1}{k_c} \log \frac{e^{-||R^c \cdot \mathbf{a}^c - \mathbf{b}^c||^2}}{[\sum_{\mathbf{v}_i \in \mathcal{V}} e^{-||R^c \cdot \mathbf{a}^c - \mathbf{v}_i||^2}]^\alpha} \tag{2}$$

During learning this function $G$ (for Goodness) is maximized with respect to all the vector and matrix components. This gives a much better generalization performance than the one obtained by just maximizing $D$. The results presented in the next sections were obtained by maximizing $G$ using gradient ascent. All the vector and matrix components were updated simultaneously at each iteration. One effective method of performing the optimization is conjugate gradient. Learning was fast, usually requiring only a few hundred updates. It is worth pointing out that, in general, different initial configurations and optimization algorithms caused the system to arrive at different solutions, but these solutions were almost always very similar in terms of generalization performance.

## 2 LRE results

Here we present the results obtained applying LRE to the Family Tree Problem [1]. In this problem, the data consists of people and relations among people belonging to two families, one Italian and one English, shown in fig.1 (left) [5]. All the information in these trees can be represented in simple propositions of the form $(person_1, relation, person_2)$. Using the relations *father, mother, husband, wife, son, daughter, uncle, aunt, brother, sister, nephew, niece* there are 112 such triplets in the two trees. Fig.1 (right) shows the embedding obtained after training with LRE. Notice how the Italians are linearly separable from the English people. From the Hinton diagram, we can see that each member of a family is symmetric to the corresponding member in the other family. The sign of the third component of the vectors is (almost) a feature for the nationality. When testing the generalization

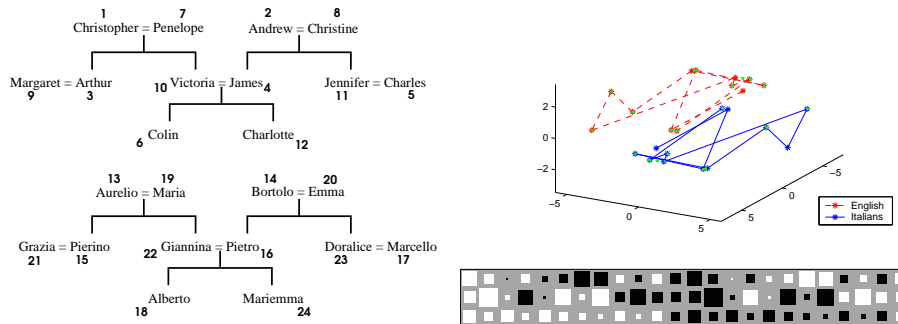

Figure 1: Left: Two isomorphic family trees. The symbol "=" means "married to". Right Top: layout of the vectors representing the people obtained for the Family Tree Problem in 3D. Vectors end-points are indicated by *, the ones in the same family tree are connected to each other. All 112 triplets were used for training. Right Bottom: Hinton diagrams of the 3D vectors shown above. The vector of each person is a column, ordered according to the numbering on the tree diagram on the left.

performance, for each triplet in the test set $(\mathbf{a}, R, ?)$, we chose as completion the concepts $\mathbf{k}$ according to their probability, given $R^c \cdot \mathbf{a}^c$. The system was generally able to complete correctly all 112 triplets even when 28 of them, picked at random, had been left out during training. These results on the Family Tree Problem are much better than the ones obtained using any other method on the same problem: Quinlan's FOIL [7] could generalize on 4 triplets, while Hinton (1986) and O'Reilly (1996) made one or more errors when only 4 test cases were held out during training.

For most problems there exist triplets which cannot be completed. This is the case, for example, of (*Christopher, father, ?*) in the Family Tree Problem. Therefore, here we argue that it is not sufficient to test generalization by merely testing the completion of those complete-able triplets which have not been used for training. The proper test for generalization is to see how the system completes *any* triplet of the kind $(\mathbf{a}, R, ?)$ where $\mathbf{a}$ ranges over the concepts and R over the relations. We cannot assume to have knowledge of which triplets admit a completion, and which do not. To our knowledge this issue has never been analyzed before (even though FOIL handles this problem correctly). To do this the system needs a way to indicate when a triplet does not admit a completion. Therefore, once the maximization of $G$ is terminated, we build a new probabilistic model around the solution which has been found. This new model is constituted, for each relation, of a mixture of $N$ identical spherical Gaussians, each centered on a concept vector, and a Uniform distribution. The Uniform distribution will take care of the "don't know" answers, and will be competing with all the other Gaussians, each representing a concept vector. For each relation the Gaussians have different variances and the Uniform a different height. The parameters of this probabilistic model are, for each relation $R$, the variances of the Gaussians $\sigma_R$ and the relative density under the Uniform distribution, which we shall write as $\exp(-r_R^2/2\sigma_R^2)$. These parameters are learned using a validation set, which will be the union of a set of complete-able (positive) triplets $\mathcal{P}$ and a set of pairs which cannot be completed $\mathcal{N}$ (negative); that is $\mathcal{P} = \{\mathbf{a}^p, R^p, \mathbf{b}^p\}_{p=1}^P$ and $\mathcal{N} = \{\mathbf{a}^q, R^q, \bot\}_{q=1}^Q$ where $\bot$ indicates the fact that the result of applying relation $R^q$ to $\mathbf{a}^q$ does not belong to $\mathcal{V}$. This is done by maximizing the following discriminative goodness function $F$ over the validation set :

$$
\begin{aligned}
F \quad = \quad & \sum_{q=1}^{Q} \log \frac{\exp(-\frac{r_R^2}{2\sigma_R^2})}{\exp(-\frac{r_R^2}{2\sigma_R^2}) + \sum_{\mathbf{v}_i \in \mathcal{V}} \exp(-\frac{\|R^q \cdot \mathbf{a}^q - \mathbf{v}_i\|^2}{2\sigma_R^2})} \\
+ \quad & \sum_{p=1}^{P} \frac{1}{k_p} \cdot \log \frac{\exp(-\frac{\|R^p \cdot \mathbf{a}^p - \mathbf{b}^p\|^2}{2\sigma_R^2})}{\exp(-\frac{r_R^2}{2\sigma_R^2}) + \sum_{\mathbf{v}_i \in \mathcal{V}} \exp(-\frac{\|R^p \cdot \mathbf{a}^p - \mathbf{v}_i\|^2}{2\sigma_R^2})}
\end{aligned}
\tag{3}
$$

with respect to the $\sigma_R$ and $r_R$ parameters, while everything else is kept fixed. Having learned these parameters, in order to complete any triplet $(R, \mathbf{a}, ?)$ we compute the probability distribution over each of the Gaussians and the Uniform distribution given $R \cdot \mathbf{a}$. The system then chooses a vector $\mathbf{v}_i$ or the "don't know" answer according to those probabilities, as the completion to the triplet.

We used this method on the Family Tree Problem using a train, test and validation sets built in the following way. The test set contained 12 positive triplets chosen at random, but such that there was a triplet per relation. The validation set contained a group of 12 positive and a group of 12 negative triplets, chosen at random and such that each group had a triplet per relation. The train set contained the remaining 88 positive triplets. After learning a distributed representation for the entities in the data by maximizing $G$ over the training set, we learned the parameters of the probabilistic model by maximizing $F$ over the validation set. The resulting system was able to correctly complete all the 288 possible triplets $(R, \mathbf{a}, ?)$. Figure 2 shows the distribution of the probabilities when completing one complete-able and one uncomplete-able triplet in the test set.

LRE seems to scale up well to problems of bigger size. We have used it on a much bigger version of the Family Tree Problem, where the family tree is a branch of the real family tree of one of the authors containing 49 people over 5 generations. Using the same set of 12 relations used in the Family Tree Problem, there is a total of 644 positive triplets. After learning using a training set of 524 positive triplets, and a validation set constituted

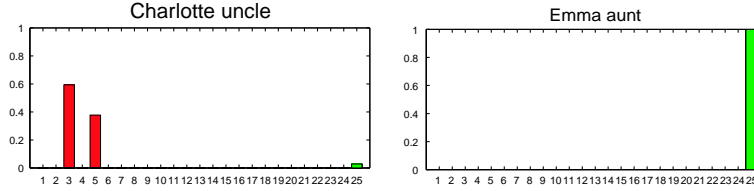

Figure 2: Distribution of the probabilities assigned to each concept for one complete-able (left) and one uncomplete-able (right) triplet written above each diagram. The complete-able triplet has two correct completions but neither of the triplets had been used for training. Black bars from 1 to 24 are the probabilities of the people ordered according to the numbering in fig.1. The last grey bar on the right, is the probability of the "don't know" answer.

by 30 positive and 30 negative triplets, the system is able to complete correctly almost all the possible triplets. When many completions are correct, a high probability is always assigned to each one of them. Only in few cases is a non-negligible probability assigned to some wrong completions. Almost all the generalization errors are of a specific form. The system appears to believe that "brother/sister of" means "son/daughter of parents of". It fails to model the extra restriction that people cannot be their own brother/sister. On the other hand, nothing in the data specifies this restriction.

## 3 Using LRE to represent recursive data structures

In this section, we shall show how LRE can be used effectively to find compact distributed representations for variable-sized recursive data structures, such as trees and lists. Here we discuss binary trees, but the same reasoning applies to trees of any valence. The approach is inspired by Pollack's RAAM architecture [6]. A RAAM is an auto-encoder which is trained using backpropagation. Figure 3 shows the architecture of the network for binary trees. The system can be thought as being composed of two networks. The first one, called

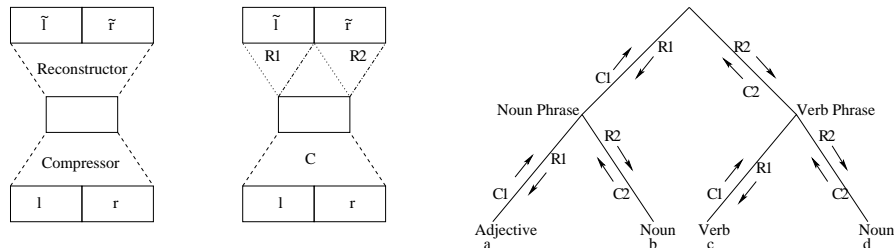

Figure 3: Left: the architecture of a RAAM for binary trees. The 3 layers are fully connected. Adapted from [6]. Center: how LRE can be used to learn a representation for binary trees in a RAAM-like fashion. Right: the binary tree structure of the sentences used in the experiment.

*compressor* encodes two fixed-width patterns into a single pattern of the same size. The second one, called *reconstructor*, decodes a compressed pattern into facsimiles of its parts, and determines when the parts should be further decoded. To encode a tree the network must learn as many auto-associations as the total number of non-terminal nodes in the tree. The codes for the terminal nodes are supplied, and the network learns suitable codes for the other nodes. The decoding procedure must decide whether a decoded vector represents

a terminal node or an internal node which should be further decoded. This is done by using binary codes for the terminal symbols, and then fixing a threshold which is used for checking for "binary-ness" during decoding.

The RAAM approach can be cast as an LRE problem, in which concepts are trees, sub-trees or leaves, or pairs of trees, sub-trees or leaves, and there exist 3 relationships: $C$ implementing the compressor, and $R_1$ and $R_2$ which jointly implement the reconstructor (see fig.3). We can then learn a representation for all the trees, and the matrices by maximizing $G$ in eq.2. This formulation, which we have called Hierarchical LRE (HLRE), solves two problems encountered in RAAMs. First, one does not need to supply codes for the leaves of the trees, since LRE will learn an appropriate distributed representation for them. Secondly, one can also learn from the data when to stop the decoding process. In fact, the problem of recognizing whether a node needs to be further decoded, is similar to the problem of recognizing that a certain triplet does not admit a completion, that we solved in the previous section. While before we built an outlier model for the "don't know" answers, now we shall build one for the non-terminal nodes. This can be done by learning appropriate values of $\sigma$ and $r$ for relations $R_1$ and $R_2$ maximizing $F$ in eq.3. The set of triplets $(\mathbf{a}^c, R^c, \mathbf{b}^c)$ where $\mathbf{b}^c$ is not a leaf of the tree, will play the role of the $\mathcal{N}$ set which appears in eq.3.

We have applied this method to the problem of encoding binary trees which correspond to sentences of 4 words from a small vocabulary. Sentences had a fixed structure: a noun phrase, constituted of an adjective and a noun, followed by a verb phrase, made of a verb and a noun (see fig.3). Thus each sentence had a fixed grammatical structure, to which we added some extra semantic structure in the following way. Words of each grammatical category were divided into two disjoint sets. Nouns were in $N_\alpha = \{girl, woman, scientist\}$ or in $N_\beta = \{dog, doctor, lawyer\}$; adjectives were in $A_\alpha = \{pretty, young\}$ or in $A_\beta = \{ugly, old\}$; verbs were in $V_\alpha = \{help, love\}$ or in $V_\beta = \{hurt, annoy\}$. Our training set was constituted by 10 sentences of the type: $(adjective_\alpha, noun_\alpha, verb_\alpha, noun_{\alpha \cup \beta})$ and 10 of the type $(adjective_\beta, noun_\beta, verb_\beta, noun_{\alpha \cup \beta})$, where the suffix indicates the set to which each word type belongs. In this way, sentences of the kind "pretty girl annoy scientist" were not allowed in the training set, and there were 144 possible sentences that satisfied the constraints which were implicit in the training set.

We used HLRE to learn a distributed representation for all the nodes in the trees, maximizing $G$ using the 20 sentences in the training set. In 7D, after having built the outlier model for the non-terminal symbols, given any root or internal node the system would reconstruct its children, and if they were non-terminal symbols would further decode each of them. The decoding process would always halt providing the correct reconstruction for all the 20 sentences in the training set. The top row of fig.4 shows the distributed representations found for each word in the vocabulary. Notice how the $\alpha$ and $\beta$ sets of adjectives and verbs are almost symmetric with respect to the origin; the difference between the $\alpha$ and $\beta$ sets is less evident for the nouns, due to the fact that while there exists a restriction on which nouns can be used in position $b$, there is no restriction on the nouns appearing in position $d$ in the training sentences (see fig.3, right). We tested how well this system could generalize beyond the training set using the same procedure used by Pollack to enumerate the set of trees that RAAMs are able to represent [6]: for every pair of patterns for trees, first we encoded them into a pattern for a new higher level tree, and then we decoded this tree back into the patterns of the two sub-trees. If the norm of the difference between the original and the reconstructed sub-trees was within a tolerance, which we set to 0.1, then the tree could be considered to be well formed. The system shows impressive generalization performance: after training using the 20 sentences, the four-word sentences it generates are all the 144 well formed sentences, and only those. It does not generate sentences which are either grammatically wrong, like "dog old girl annoy", nor sentences which violate semantic constraints, like "pretty girl annoy scientist". This is striking when compared to the poor generalization performance obtained by the RAAM on similar problems. As recognized by

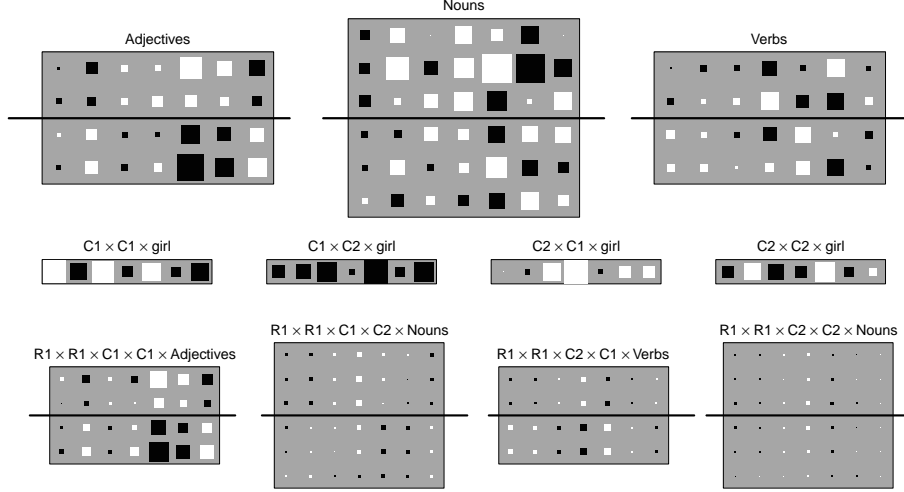

Figure 4: For Hinton diagrams with multiple rows, each row relates to a word, in the following order - Adjectives:{1=pretty; 2=young; 3=ugly; 4=old}; Nouns:{1=girl; 2=woman; 3=scientist; 4=dog; 5=doctor; 6=lawyer}; Verbs:{1=help; 2=love; 3=hurt; 4=annoy}; . Black bars separate $A_\alpha$, $N_\alpha$, $V_\alpha$ (higher), from $A_\beta$, $N_\beta$, $V_\beta$ (lower). Top row: The distributed representation of the words in the sentences found after learning. Center row: The different contributions given to the root of the tree by the word "girl" when placed in position $a$, $b$, $c$ and $d$ in the tree. Bottom row: The contribution of each leaf to the reconstruction of $a$, when adjectives, nouns, verbs and nouns are applied in positions $a$, $b$, $c$ and $d$ respectively.

Pollack [6], this was almost certainly due to the fact that for the RAAMs the representation for the leaves was too similar, a problem that the HLRE formulation solves, since it learns their distributed representations.

Let us try to explain why HLRE can generalize so well. The $C$ matrix can be decomposed into two sub-matrices, $C_1$ and $C_2$, such that for any two children of a given node, $l$ and $r$, we have: $C \cdot [l; r] = C_1 \cdot l + C_2 \cdot r$, where ";" denotes the concatenation operator. Therefore we have a pair of matrices, either $(C_1, R_1)$ or $(C_2, R_2)$, associated to each link in the graph. Once the system has learned an embedding, finding a distributed representation for a given tree amounts to multiplying the representation of its leaves by all the $C_i$ matrices found on all the paths from the leaves to the root, and adding them up. Luckily matrix multiplication is non-commutative, and therefore every sequence of words on its leaves can generate a different representation at the root node. The second row of fig.4 makes this point clear showing the different contributions given to the root of the tree by the word "girl", depending on its position in the sentence. A tree can be "unrolled" from the root to its leaves by multiplying its distributed representation using the $R_i$ matrices. We can now analyze how a particular leaf is reconstructed. Leaf $a$, for example, is reconstructed as:

$$\tilde{a} = R_1^2 \cdot C_1^2 \cdot a + R_1^2 \cdot C_1 \cdot C_2 \cdot b + R_1^2 \cdot C_2 \cdot C_1 \cdot c + R_1^2 \cdot C_2^2 \cdot d$$

The third row of fig.4 shows the contribution of each leaf to the reconstruction of $a$, when adjectives, nouns, verbs and nouns are placed on leaves $a$, $b$, $c$ and $d$ respectively. We can see that the contributions from the adjectives, match very closely their actual distributed representations, while the contributions from the nouns in position $d$ are negligible. This means that any adjective placed on $a$ will tend to be reconstructed correctly, and that its reconstruction is independent of the noun we have in position $d$. On the other hand, the

contributions from nouns and verbs in positions $b$ and $c$ are non-negligible, and notice how those given by words belonging to the $\alpha$ subsets are almost symmetric to those given by words in the $\beta$ subsets. In this way the system is able to enforce the semantic agreement between words in positions $a$, $b$ and $c$. Finally, the reconstruction of $a$, when adjectives, nouns, verbs and nouns are not placed on leaves $a$, $b$, $c$ and $d$ respectively, assigns a very low probability to any word, and thus the system does not generate sentences which are not well formed.

## 4    Conclusions

Linear Relational Embedding is a new method for learning distributed representations of concepts and relations from data consisting of instances of relations between given concepts. It finds a mapping from the concepts into a feature-space by imposing the constraint that relations in this feature-space are modeled by linear operations. LRE shows excellent generalization performance. The results on the Family Tree Problem are far better than those obtained by any previously published method. Results on other problems are similar. Moreover we have shown elsewhere [4] that, after learning a distributed representation for a set of concepts and relations, LRE can easily modify these representations to incorporate new concepts and relations and that it is possible extract logical rules from the solution and to couple LRE with FOIL [7]. Learning is fast and LRE rarely converges to solutions with poor generalization. We began introducing LRE for binary relations, and then we saw how these ideas can be easily extended to higher arity relation by simply concatenating concept vectors and using rectangular matrices for the relations. The compressor relation for binary trees is a ternary relation; for trees of higher valence the compressor relation will have higher arity. We have seen how HLRE can be used to find distributed representations for hierarchical structures, and its generalization performance is much better than the one obtained using RAAMs on similar problems.

It is easy to prove that, when all the relations are binary, given a sufficient number of dimensions, there always exists an LRE-type of solution that satisfies any set of triplets [4]. However, due to its linearity, LRE cannot represent some relations of arity greater than 2. This limitation can be overcome by adding an extra layer of non-linear units for representing the relations. This new method, called Non-Linear Relational Embedding (NLRE) [4], can represent any relation and has given good generalization results.

## Footnotes

[1]Methods analogous to the ones presented here that can be used to complete any element of a triplet can be found in [4].

[2] We would like our system to assign equal probability to each of the correct completions. The discrete probability distribution that we want to approximate is therefore: $\mathsf{P}_\mathbf{x} = \frac{1}{d} \sum_{i=1}^{d} \delta(\mathbf{b}_i - \mathbf{x})$ where $\delta$ is the discrete delta function and $\mathbf{x}$ ranges over the vectors in $\mathcal{V}$. Our system implements the discrete probability distribution: $\mathsf{Q}_\mathbf{x} = \frac{1}{Z} \exp(-||R \cdot \mathbf{a} - \mathbf{x}||^2)$ where $Z$ is the normalization factor. The $1/k_c$ factor in eq.1 ensures that we are minimizing the Kullback-Leibler divergence between $\mathsf{P}$ and $\mathsf{Q}$.

[3] The obvious approach to find an embedding would be to minimize the sum of squared distances between $R^c \cdot \mathbf{a}^c$ and $\mathbf{b}^c$ over all the triplets, with respect to all the vector and matrix components. Unfortunately this minimization (almost) always causes all of the vectors and matrices to collapse to the trivial $\mathbf{0}$ solution.

[4] For one-to-many relations we must not decrease the value of $\alpha$ all the way to 0, because this would cause some concept vectors to become coincident. This is because the only way to make $R^c \cdot \mathbf{a}^c$ equal to $k_c$ different vectors, is by collapsing them onto a unique vector.

[5]The names of the Italian family have been altered from those originally used in Hinton (1986) to match those of one of the author's family.

## References

[1]  Geoffrey E. Hinton. Learning distributed representations of concepts. In *Proceedings of the Eighth Annual Conference of the Cognitive Science Society*, pages 1–12. Erlbaum, NJ, 1986.

[2]  Geoffrey E. Hinton, James L. McClelland, and David E. Rumelhart. Distributed representations. In David E. Rumelhart, James L. McClelland, and the PDP research Group, editors, *Parallel Distributed Processing*, volume 1, pages 77–109. The MIT Press, 1986.

[3]  Randall C. O'Reilly. *The LEABRA model of neural interactions and learning in the neocortex*. PhD thesis, Department of Psychology, Carnegie Mellon University, 1996.

[4]  Alberto Paccanaro. *Learning Distributed Representations of Relational Data using Linear Relational Embedding*. PhD thesis, Computer Science Department, University of Toronto, 2002.

[5]  Alberto Paccanaro and Geoffrey E. Hinton. Learning distributed representations by mapping concepts and relations into a linear space. In Pat Langley, editor, *Proceedings of ICML2000*, pages 711–718. Morgan Kaufmann, Stanford University, 2000.

[6]  Jordan B. Pollack. Recursive distributed representations. *Artificial Intelligence*, 46:77–105, 1990.

[7]  J. R. Quinlan. Learning logical definitions from relations. *Machine Learning*, 5:239–266, 1990.
